# Layer-wise analysis of deep networks with Gaussian kernels

**Grégoire Montavon**
Machine Learning Group
TU Berlin
gmontavon@cs.tu-berlin.de

**Mikio L. Braun**
Machine Learning Group
TU Berlin
mikio@cs.tu-berlin.de

**Klaus-Robert Müller**
Machine Learning Group
TU Berlin
krm@cs.tu-berlin.de

## Abstract

Deep networks can potentially express a learning problem more efficiently than local learning machines. While deep networks outperform local learning machines on some problems, it is still unclear how their nice representation emerges from their complex structure. We present an analysis based on Gaussian kernels that measures how the representation of the learning problem evolves layer after layer as the deep network builds higher-level abstract representations of the input. We use this analysis to show empirically that deep networks build progressively better representations of the learning problem and that the best representations are obtained when the deep network discriminates only in the last layers.

## 1 Introduction

Local learning machines such as nearest neighbors classifiers, radial basis function (RBF) kernel machines or linear classifiers predict the class of new data points from their neighbors in the input space. A limitation of local learning machines is that they cannot generalize beyond the notion of continuity in the input space. This limitation becomes detrimental when the Bayes classifier has more variations (ups and downs) than the number of labeled samples available. This situation typically occurs on problems where an instance — let's say, a handwritten digit — can take various forms due to irrelevant variation factors such as its position, its size, its thickness and more complex deformations. These multiple factors of variation can greatly increase the complexity of the learning problem (Bengio, 2009).

This limitation motivates the creation of learning machines that can map the input space into a higher-level representation where regularities of higher order than simple continuity in the input space can be expressed. Engineered feature extractors, nonlocal kernel machines (Zien et al., 2000) or deep networks (Rumelhart et al., 1986; LeCun et al., 1998; Hinton et al., 2006; Bengio et al., 2007) can implement these more complex regularities. Deep networks implement them by distorting the input space so that initially distant points in the input space appear closer. Also, their multilayered nature acts as a regularizer, allowing them to share at a given layer features computed at the previous layer (Bengio, 2009). Understanding how the representation is built in a deep network and how to train it efficiently received a lot of attention (Goodfellow et al., 2009; Larochelle et al., 2009; Erhan et al., 2010). However, it is still unclear how their nice representation emerges from their complex structure, in particular, how the representation evolves from layer to layer.

The main contribution of this paper is to introduce an analysis based on RBF kernels and on the kernel principal component analysis (kPCA, Schölkopf et al., 1998) that can capture and quantify the layer-wise evolution of the representation in a deep network. In practice, for each layer $1 \leq l \leq L$ of the deep network, we take a small labeled dataset $\mathcal{D}$, compute its image $\mathcal{D}^{(l)}$ at the layer $l$ of the deep network and measure what dimensionality the local model built on top of $\mathcal{D}^{(l)}$ must have in order to solve the learning problem with a certain accuracy.

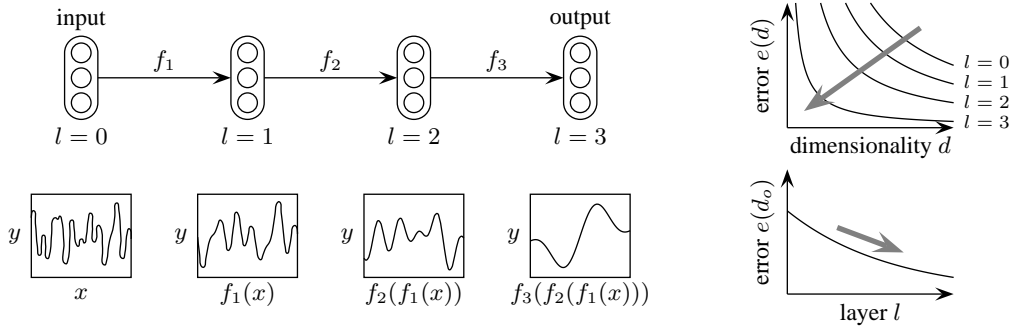

Figure 1: As we move from the input to the output of the deep network, better representations of the learning problem are built. We measure this improvement with the layer-wise RBF analysis presented in Section 2 and Section 3.2. This analysis relates the prediction error $e(d)$ to the dimensionality $d$ of a local model built at each layer of the deep network. As the data is propagated through the deep network, lower errors are obtained with lower-dimensional local models. The plots on the right illustrate this dynamic where the thick gray arrows indicate the forward path of the deep network and where $d_o$ is a fixed number of dimensions.

We apply this novel analysis to a multilayer perceptron (MLP), a pretrained multilayer perceptron (PMLP) and a convolutional neural network (CNN). We observe in each case that the error and the dimensionality of the local model decrease as we propagate the dataset through the deep network. This reveals that the deep network improves the representation of the learning problem layer after layer. This progressive layer-wise simplification is illustrated in Figure 1. In addition, we observe that the CNN and the PMLP tend to postpone the discrimination to the last layers, leading to more transferable features and better-generalizing representations than for the simple MLP. This result suggests that the structure of a deep network, by enforcing a separation of concerns between low-level generic features and high-level task-specific features, has an important role to play in order to build good representations.

## 2    RBF analysis of a learning problem

We would like to quantify the complexity of a learning problem $p(y \mid x)$ where samples are drawn independently from a probability distribution $p(x, y)$. A simple way to do it is to measure how many degrees of freedom (or dimensionality $d$) a local model must have in order to solve the learning problem with a certain error $e$. This analysis relates the dimensionality $d$ of the local model to its prediction error $e(d)$.

In practice, there are many ways to define the dimensionality of a model, for example, (1) the number of samples given to the learning machine, (2) the number of required hidden nodes of a neural network (Murata et al., 1994), (3) the number of support vectors of a SVM or (4) the number of leading kPCA components of the input distribution $p(x)$ used in the model. The last option is chosen for the following two reasons:

First, the kPCA components are added cumulatively to the prediction model as the dimensionality of the model increases, thus offering stability, while in the case of support vector machines, previously chosen support vectors might be dropped in favor of other support vectors in higher-dimensional models.

Second, the leading kPCA components obtained with a finite and typically small number of samples $n$ are similar to those that would be obtained in the asymptotic case where $p(x, y)$ is fully observed ($n \rightarrow \infty$). This property is shown by Braun (2006) and Braun et al. (2008) in the case of a single kernel, and by extension, in the case of a finite set of kernels.

This last property is particularly useful since $p(x, y)$ is unknown and only a finite number of observations are available. The analysis presented here is strongly inspired from the relevant dimensionality estimation (RDE) method of Braun et al. (2008) and is illustrated in Figure 2 for a small two-

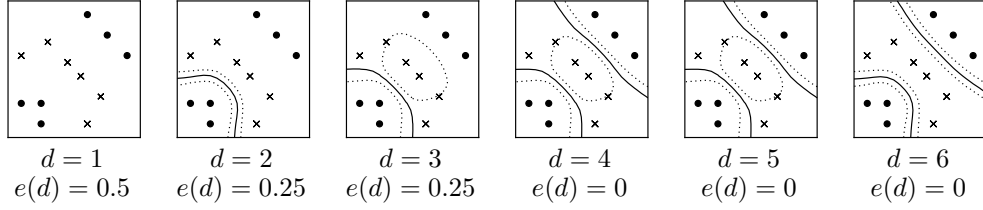

Figure 2: Illustration of the RBF analysis on a toy dataset of 12 samples. As we add more and more leading kPCA components, the model becomes more flexible, creating a better decision boundary. Note that with four leading kPCA components out of the 12 kPCA components, all the samples are already classified perfectly.

dimensional toy example. In the next lines, we present the computation steps required to estimate the error as a function of the dimensionality.

Let $\{(x_1, y_1), \ldots, (x_n, y_n)\}$ be a dataset of $n$ points drawn independently from $p(x, y)$ where $y_i$ is an indicator vector having value 1 at the index corresponding to the class of $x_i$ and 0 elsewhere. Let $X = (x_1, \ldots, x_n)$ and $Y = (y_1, \ldots, y_n)$ be the matrices associated to the inputs and labels of the dataset. We compute the kernel matrix $K$ associated to the dataset:

$$[K]_{ij} = k(x_i, x_j) \qquad \text{where} \quad k(x, x') = \exp\left(-\frac{\|x - x'\|^2}{2\sigma^2}\right) \quad .$$

The kPCA components $u_1, \ldots, u_n$ are obtained by performing an eigendecomposition of $K$ where eigenvectors $u_1, \ldots, u_n$ have unit length and eigenvalues $\lambda_1, \ldots, \lambda_n$ are sorted by decreasing magnitude:

$$K = (u_1 | \ldots | u_n) \cdot \text{diag}(\lambda_1, \ldots, \lambda_n) \cdot (u_1 | \ldots | u_n)^\top$$

Let $\hat{U} = (u_1 | \ldots | u_d)$ and $\hat{\Lambda} = \text{diag}(\lambda_1, \ldots, \lambda_d)$ be a d-dimensional approximation of the eigendecomposition. We fit a linear model $\beta^\star$ that maps the projection on the $d$ leading components of the training data to the log-likelihood of the classes

$$\beta^\star = \text{argmin}_\beta \|\exp(\hat{U}\hat{U}^\top \beta) - Y\|_F^2$$

where $\beta$ is a matrix of same size as $Y$ and where the exponential function is applied element-wise. The predicted class log-probability $\log(\hat{y})$ of a test point $(x, y)$ is computed as

$$\log(\hat{y}) = k(x, X)\hat{U}\hat{\Lambda}^{-1}\hat{U}^\top \beta^\star + C$$

where $k(x, X)$ is a matrix of size $1 \times n$ computing the similarities between the new point and each training point and where $C$ is a normalization constant. The test error is defined as:

$$e(d) = \Pr(\text{argmax}\,\hat{y} \neq \text{argmax}\,y)$$

The training and test error can be used as an approximation bound for the asymptotic case $n \to \infty$ where the data would be projected on the real eigenvectors of the input distribution. In the next sections, the training and test error are depicted respectively as dotted and solid lines in Figure 3 and as the bottom and the top of error bars in Figure 4. For each dimension, the kernel scale parameter $\sigma$ that minimizes $e(d)$ is retained, leading to a different kernel for each dimensionality. The rationale for taking a different kernel for each model is that the optimal scale parameter typically shrinks as more leading components of the input distribution are observed.

## 3 Methodology

In order to test our two hypotheses (the progressive emergence of good representations in deep networks and the role of the structure for postponing discrimination), we consider three deep networks of interest, namely a convolutional neural network (CNN), a multilayer perceptron (MLP) and a variant of the multilayer perceptron pretrained in an unsupervised fashion with a deep belief

network (PMLP). These three deep networks are chosen in order to evaluate how the two types of regularizers implemented respectively by the CNN and the PMLP impact on the evolution of the representation layer after layer. We describe how they are built, how they are trained and how they are analyzed layer-wise with the RBF analysis described in Section 2.

The *multilayer perceptron* (MLP) is a deep network obtained by alternating linear transformations and element-wise nonlinearities. Each layer maps an input vector of size $m$ into an output vector of size $n$ and consists of (1) a linear transformation $\text{linear}_{m \to n}(x) = w \cdot x + b$ where $w$ is a weight matrix of size $n \times m$ learned from the data and (2) a non-linearity applied element-wise to the output of the linear transformation. Our implementation of the MLP maps two-dimensional images of $28 \times 28$ pixels into a vector of size 10 (the 10 possible digits) by applying successively the following functions:

$$f_1(x) = \tanh(\text{linear}_{28 \times 28 \to 784}(x))$$
$$f_2(x) = \tanh(\text{linear}_{784 \to 784}(x))$$
$$f_3(x) = \tanh(\text{linear}_{784 \to 784}(x))$$
$$f_4(x) = \text{softmax}(\text{linear}_{784 \to 10}(x))$$

The *pretrained multilayer perceptron* (Hinton et al., 2006) that we abbreviate PMLP in this paper is a variant of the MLP where weights are initialized with a deep belief network (DBN, Hinton et al., 2006) using an unsupervised greedy layer-wise pretraining procedure. This particular weight initialization acts as a regularizer, allowing to learn better-generalizing representation of the learning problem than the simple MLP.

The *convolutional neural network* (CNN, LeCun et al., 1998) is a deep network obtained by alternating *convolution filters* $y = \text{convolve}_{m \to n}^{a \times b}(x)$ transforming a set of $m$ input features maps $\{x_1, \ldots, x_m\}$ into a set of $n$ output features maps $\{y_i = \sum_{j=1}^{m} w_{ij} \star x_j + b_i , \ i = 1 \ldots, n\}$ where the convolution filters $w_{ij}$ of size $a \times b$ are learned from data, and *pooling units* subsampling each feature map by a factor two. Our implementation maps images of $32 \times 32$ pixels into a vector of size 10 (the 10 possible digits) by applying successively the following functions:

$$f_1(x) = \tanh(\text{pool}(\text{convolve}_{1 \to 36}^{5 \times 5}(x)))$$
$$f_2(x) = \tanh(\text{pool}(\text{convolve}_{36 \to 36}^{5 \times 5}(x)))$$
$$f_3(x) = \tanh(\text{linear}_{5 \times 5 \times 36 \to 400}(x))$$
$$f_4(x) = \text{softmax}(\text{linear}_{400 \to 10}(x))$$

The CNN is inspired by the structure of biological visual systems (Hubel and Wiesel, 1962). It combines three ideas into a single architecture: (1) only local connections between neighboring pixels are allowed, (2) the convolution operator applies the same filter over the whole feature map and (3) a pooling mechanism at the top of each convolution filter adds robustness to input distortion. These mechanisms act as a regularizer on images and other types of sequential data, and learn well-generalizing models from few data points.

### 3.1 Training the deep networks

Each deep network is trained on the MNIST handwriting digit recognition dataset (LeCun et al., 1998). The MNIST dataset consists of predicting the digit $0 - 9$ from scanned handwritten digits of $28 \times 28$ pixels. We partition randomly the MNIST training set in three subsets of 45000, 5000 and 10000 samples that are respectively used for training the deep network, selecting the parameters of the deep network and performing the RBF analysis.

We consider three training procedures:

1. *No training*: the weights of the deep network are left at their initial value. If the deep network hasn't received unsupervised pretraining, the weights are set randomly according to a normal distribution $\mathcal{N}(0, \gamma^{-1})$ where $\gamma$ denotes for a given layer the number of input nodes that are connected to a single output node.

2. *Training on an alternate task*: the deep network is trained on a binary classification task that consists of determining whether the digit is original (positive example) or whether it has

been transformed by one of the 11 possible rotation/flip combinations that differs from the original (negative example). This problem has therefore 540000 labeled samples (45000 positives and 495000 negatives). The goal of training a deep network on an alternate task is to learn features on a problem where the number of labeled samples is abundant and then reuse these features to learn the target task that has typically few labels. In the alternate task described earlier, negative examples form a cloud around the manifold of positive examples and learning this manifold potentially allows the deep network to learn features that can be transfered to the digit recognition task.

3. *Training on the target task*: the deep network is trained on the digit recognition task using the 45000 labeled training samples.

These procedures are chosen in order to assess the forming of good representations in deep networks and to test the role of the structure of deep networks on different aspects of learning, such as the effectiveness of random projections, the transferability of features from one task to another and the generalization to new samples of the same distribution.

## 3.2  Applying the RBF analysis to deep networks

In this section, we explain how the RBF analysis described in Section 2 is applied to analyze layer-wise the deep networks presented in Section 3.

Let $f = f_L \circ \cdots \circ f_1$ be the trained deep network of depth $L$. Let $\mathcal{D}$ be the analysis dataset containing the 10000 samples of the MNIST dataset on which the deep network hasn't been trained. For each layer, we build a new dataset $\mathcal{D}^{(l)}$ corresponding to the mapping of the original dataset $\mathcal{D}$ to the $l$ first layers of the deep network. Note that by definition, the index zero corresponds to the raw input data (mapped through zero layers):

$$\mathcal{D}^{(l)} = \begin{cases} \mathcal{D} & l = 0 \\ \{(f_l \circ \cdots \circ f_1(x), t) \mid (x, t) \in \mathcal{D}\} & 1 \leq l \leq L \end{cases} .$$

Then, for each dataset $\mathcal{D}^{(0)}, \ldots, \mathcal{D}^{(L)}$ we perform the RBF analysis described in Section 2. We use $n = 2500$ samples for computing the eigenvectors and the remaining 7500 samples to estimate the prediction error of the model. This analysis yields for each dataset $\mathcal{D}^{(l)}$ the error as a function of the dimensionality of the model $e(d)$. A typical evolution of $e(d)$ is depicted in Figure 1.

The goal of this analysis is to observe the evolution of $e(d)$ layer after layer for the deep networks and training procedures presented in Section 3 and to test the two hypotheses formulated in Section 1 (the progressive emergence of good representations in deep networks and the role of the structure for postponing discrimination). The interest of using a local model to solve the learning problem is that the local models are blind with respect to possibly better representations that could be obtained in previous or subsequent layers. This local scoping property allows for fine isolation of the representations in the deep network. The need for local scoping also arises when "debugging" deep architectures. Sometimes, deep architectures perform reasonably well even when the first layers do something wrong. This analysis is therefore able to detect these "bugs".

The size $n$ of the dataset is selected so that it is large enough to approximate well the asymptotic case ($n \to \infty$) but also be small enough so that computing the eigendecomposition of the kernel matrix of size $n \times n$ is fast. We choose a set of scale parameters for the RBF kernel corresponding to the $0.01, 0.05, 0.10, 0.25, 0.5, 0.75, 0.9, 0.95$ and $0.99$ quantiles of the distribution of distances between pairs of data points.

## 4  Results

Layer-wise evolution of the error $e(d)$ is plotted in Figure 3 in the supervised training case. The layer-wise evolution of the error when $d$ is fixed to 16 dimensions is plotted in Figure 4. Both figures capture the simultaneous reduction of error and dimensionality performed by the deep network when trained on the target task. In particular, they illustrate that in the last layers, a few number of dimensions is sufficient to build a good model of the target task.

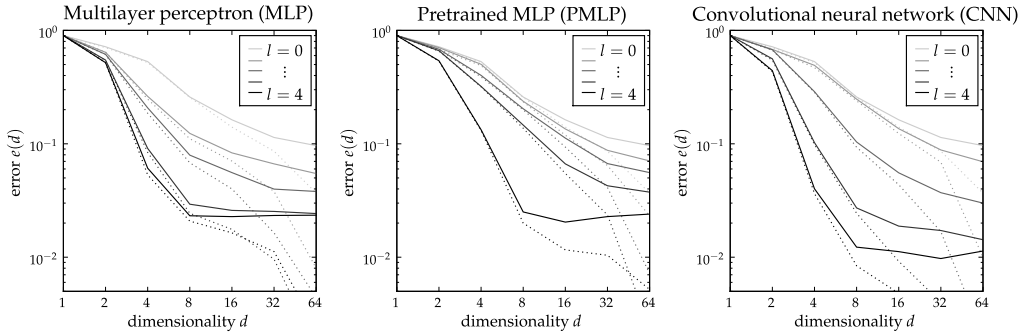

Figure 3: Layer-wise evolution of the error $e(d)$ when the deep network has been trained on the target task. The solid line and the dotted line represent respectively the test error and the training error. As the data distribution is mapped through more and more layers, more accurate and lower-dimensional models of the learning problem can be obtained.

From these results, we first demonstrate some properties of deep networks trained on an "asymptotically" large number of samples. Then, we demonstrate the important role of structure in deep networks.

## 4.1 Asymptotic properties of deep networks

When the deep network is trained on the target task with an "asymptotically" large number of samples (45000 samples) compared to the number of dimensions of the local model, the deep network builds representations layer after layer in which a low number of dimensions can create more accurate models of the learning problem.

This asymptotic property of deep networks should not be thought of as a statistical superiority of deep networks over local models. Indeed, it is still possible that a higher-dimensional local model applied directly on the raw data performs as well as a local model applied at the output of the deep network. Instead, this asymptotic property has the following consequence:

Despite the internal complexity of deep networks a local interpretation of the representation is possible at each stage of the processing. This means that deep networks do not explode the original data distribution into a statistically intractable distribution before recombining everything at the output, but instead, apply controlled distortions and reductions of the input space that preserve the statistical tractability of the data distribution at every layer.

## 4.2 Role of the structure of deep networks

We can observe in Figure 4 (left) that even when the convolutional neural network (CNN) and the pretrained MLP (PMLP) have not received supervised training, the first layers slightly improve the representation with respect to the target task. On the other hand, the representation built by a simple MLP with random weights degrades layer after layer. This observation highlights the structural prior encoded by the CNN: by convolving the input with several random convolution filters and subsampling subsequent feature maps by a factor two, we obtain a random projection of the input data that outperforms the implicit projection performed by an RBF kernel in terms of task relevance. This observation closely relates to results obtained in (Ranzato et al., 2007; Jarrett et al., 2009) where it is observed that training the deep network while keeping random weights in the first layers still allows for good predictions by the subsequent layers. In the case of the PMLP, the successive layers progressively disentangle the factors of variation (Hinton and Salakhutdinov, 2006; Bengio, 2009) and simplify the learning problem.

We can observe in Figure 4 (middle) that the phenomenon is even clearer when the CNN and the PMLP are trained on an alternate task: they are able to create generic features in the first layers that transfer well to the target task. This observation suggests that the structure embedded in the CNN and the PMLP enforces a separation of concerns between the first layers that encode low-level features, for example, edge detectors, and the last layers that encode high-level task-specific

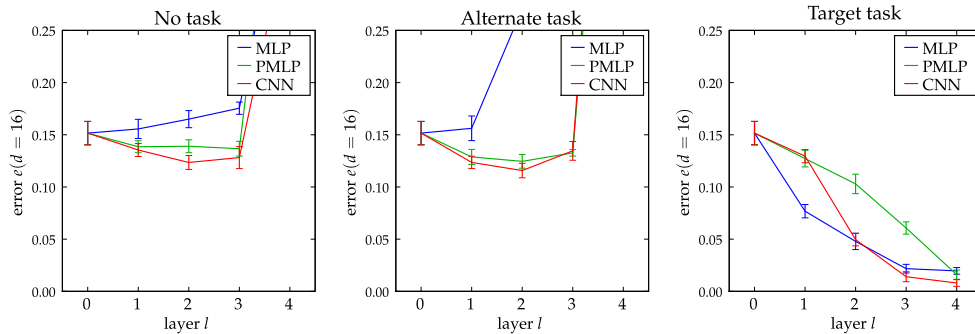

Figure 4: Evolution of the error $e(d_o)$ as a function of the layer $l$ when $d_o$ has been fixed to 16 dimensions. The top and the bottom of the error bars represent respectively the test error and the training error of the local model.

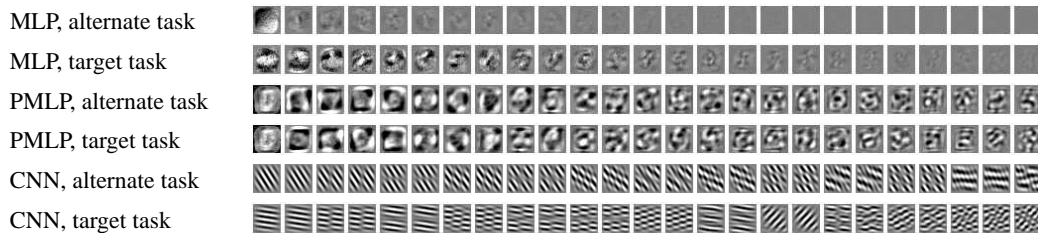

Figure 5: Leading components of the weights (receptive fields) obtained in the first layer of each architecture. The filters learned by the CNN and the pretrained MLP are richer than the filters learned by the MLP. The first component of the MLP trained on the alternate task dominates all other components and prevents good transfer on the target task.

features. On the other hand, the standard MLP trained on the alternate task leads to a degradation of representations. This degradation is even higher than in the case of random weights, despite all the prior knowledge on pixel neighborhood contained implicitly in the alternate task.

Figure 5 shows that the MLP builds receptive fields that are spatially informative but dissimilar between the two tasks. The fact that receptive fields are different for each task indicates that the MLP tries to discriminate already in the first layers. The absence of a built-in separation of concerns between low-level and high-level feature extractors seems to be a reason for the inability to learn transferable features. It indicates that end-to-end transfer learning on unstructured learning machines is in general not appropriate and supports the recent success of transfer learning on restricted portions of the deep network (Collobert and Weston, 2008; Weston et al., 2008) or on structured deep networks (Mobahi et al., 2009).

When the deep networks are trained on the target task, the CNN and the PMLP solve the problem differently as the MLP. In Figure 4 (right), we can observe that the CNN and the PMLP tend to postpone the discrimination to the last layers while the MLP starts to discriminate already in the first layers. This result suggests that again, the structure contained in the CNN and the PMLP enforces a separation of concerns between the first layers encoding low-level generic features and the last layers encoding high-level task-specific features. This separation of concerns might explain the better generalization of the CNN and PMLP observed respectively in (LeCun et al., 1998; Hinton et al., 2006). It also rejoins the findings of Larochelle et al. (2009) showing that the pretraining of the PMLP must be unsupervised and not supervised in order to build well-generalizing representations.

# 5 Conclusion

We present a layer-wise analysis of deep networks based on RBF kernels. This analysis estimates for each layer of the deep network the number of dimensions that is necessary in order to model well a learning problem based on the representation obtained at the output of this layer.

We observe that a properly trained deep network creates representations layer after layer in which a more accurate and lower-dimensional local model of the learning problem can be built.

We also observe that despite a steady improvement of representations for each architecture of interest (the CNN, the MLP and the pretrained MLP), they do not solve the problem in the same way: the CNN and the pretrained MLP seem to separate concerns by building low-level generic features in the first layers and high-level task-specific features in the last layers while the MLP does not enforce this separation. This observation emphasizes the limitations of black box transfer learning and, more generally, of black box training of deep architectures.

## References

Y. Bengio, P. Lamblin, D. Popovici, and H. Larochelle. Greedy layer-wise training of deep networks. In *Advances in Neural Information Processing Systems 19*, pages 153–160. MIT Press, 2007.

Yoshua Bengio. Learning deep architectures for AI. *Foundations and Trends in Machine Learning*, 2(1):1–127, 2009.

Mikio L. Braun. Accurate bounds for the eigenvalues of the kernel matrix. *Journal of Machine Learning Research*, 7:2303–2328, Nov 2006.

Mikio L. Braun, Joachim Buhmann, and Klaus-Robert Müller. On relevant dimensions in kernel feature spaces. *Journal of Machine Learning Research*, 9:1875–1908, Aug 2008.

R. Collobert and J. Weston. A unified architecture for natural language processing: Deep neural networks with multitask learning. In *International Conference on Machine Learning, ICML*, 2008.

Dumitru Erhan, Yoshua Bengio, Aaron C. Courville, Pierre-Antoine Manzagol, Pascal Vincent, and Samy Bengio. Why does unsupervised pre-training help deep learning? *Journal of Machine Learning Research*, 11:625–660, 2010.

Ian Goodfellow, Quoc Le, Andrew Saxe, and Andrew Y. Ng. Measuring invariances in deep networks. In *Advances in Neural Information Processing Systems 22*, pages 646–654, 2009.

G. E. Hinton and R. R. Salakhutdinov. Reducing the dimensionality of data with neural networks. *Science*, 313(5786):504–507, July 2006.

Geoffrey E. Hinton, Simon Osindero, and Yee-Whye Teh. A fast learning algorithm for deep belief nets. *Neural Comput.*, 18(7):1527–1554, 2006.

D. H. Hubel and T. N. Wiesel. Receptive fields, binocular interaction and functional architecture in the cat's visual cortex. *The Journal of physiology*, 160:106–154, January 1962.

Kevin Jarrett, Koray Kavukcuoglu, Marc'Aurelio Ranzato, and Yann LeCun. What is the best multi-stage architecture for object recognition? In *Proc. International Conference on Computer Vision (ICCV'09)*. IEEE, 2009.

Hugo Larochelle, Yoshua Bengio, Jérôme Louradour, and Pascal Lamblin. Exploring strategies for training deep neural networks. *J. Mach. Learn. Res.*, 10:1–40, 2009. ISSN 1532-4435.

Y. LeCun, L. Bottou, Y. Bengio, and P. Haffner. Gradient-based learning applied to document recognition. *Proceedings of the IEEE*, 86(1):2278–2324, November 1998.

Hossein Mobahi, Ronan Collobert, and Jason Weston. Deep learning from temporal coherence in video. In Léon Bottou and Michael Littman, editors, *Proceedings of the 26th International Conference on Machine Learning*, pages 737–744, Montreal, June 2009. Omnipress.

Noboru Murata, Shuji Yoshizawa, and Shun ichi Amari. Network information criterion - determining the number of hidden units for an artificial neural network model. *IEEE Transactions on Neural Networks*, 5:865–872, 1994.

Genevieve B. Orr and Klaus-Robert Müller, editors. *Neural Networks: Tricks of the Trade, this book is an outgrowth of a 1996 NIPS workshop*, volume 1524 of *Lecture Notes in Computer Science*, 1998. Springer.

M. A. Ranzato, Fu J. Huang, Y. L. Boureau, and Y. LeCun. Unsupervised learning of invariant feature hierarchies with applications to object recognition. In *Computer Vision and Pattern Recognition, 2007. CVPR '07. IEEE Conference on*, pages 1–8, 2007.

D. E. Rumelhart, G. E. Hinton, and R. J. Williams. Learning representations by back-propagating errors. *Nature*, 323(6088):533–536, 1986.

Bernhard Schölkopf, Alexander Smola, and Klaus-Robert Müller. Nonlinear component analysis as a kernel eigenvalue problem. *Neural Comput.*, 10(5):1299–1319, 1998.

Jason Weston, Frédéric Ratle, and Ronan Collobert. Deep learning via semi-supervised embedding. In *ICML '08: Proceedings of the 25th international conference on Machine learning*, pages 1168–1175, 2008.

Alexander Zien, Gunnar Rätsch, Sebastian Mika, Bernhard Schölkopf, Thomas Lengauer, and Klaus-Robert Müller. Engineering support vector machine kernels that recognize translation initiation sites. *Bioinformatics*, 16(9):799–807, 2000.

